# The Impact of Unlabeled Patterns in Rademacher Complexity Theory for Kernel Classifiers

**Davide Anguita, Alessandro Ghio, Luca Oneto, Sandro Ridella**
Department of Biophysical and Electronic Engineering
University of Genova
Via Opera Pia 11A, I-16145 Genova, Italy
{Davide.Anguita,Alessandro.Ghio} @unige.it
{Luca.Oneto,Sandro.Ridella} @unige.it

## Abstract

We derive here new generalization bounds, based on Rademacher Complexity theory, for model selection and error estimation of linear (kernel) classifiers, which exploit the availability of unlabeled samples. In particular, two results are obtained: the first one shows that, using the unlabeled samples, the confidence term of the conventional bound can be reduced by a factor of three; the second one shows that the unlabeled samples can be used to obtain much tighter bounds, by building localized versions of the hypothesis class containing the optimal classifier.

## 1  Introduction

Understanding the factors that influence the performance of a statistical procedure is a key step for finding a way to improve it. One of the most explored procedures in the machine learning approach to pattern classification aims at solving the well–known *model selection and error estimation* problem, which targets the estimation of the generalization error and the choice of the optimal predictor from a set of possible classifiers. For reaching this target, several approaches have been proposed [1, 2, 3, 4], which provide an upper bound on the generalization ability of the classifier, which can be used for model selection purposes as well. Typically, all these bounds consists of three terms: the first one is the empirical error of the classifier (i.e. the error performed on the training data), the second term is a bias that takes into account the complexity of the class of functions, which the classifier belongs to, and the third one is a confidence term, which depends on the cardinality of the training set. These approaches are quite interesting because they investigate the finite sample behavior of a classifier, instead of the asymptotic one, even though their practical applicability has been questioned for a long time[1]. One of the most recent methods for obtaining these bounds is to exploit the Rademacher Complexity, which is a powerful statistical tool that has been deeply investigated during the last years [5, 6, 7]. This approach has shown to be of practical use, by outperforming more traditional methods [8, 9] for model selection in the small–sample regime [10, 5, 6], i.e. when the dimensionality of the samples is comparable, or even larger, than the cardinality of the training set. We show in this work how its performance can be further improved by exploiting some extra knowledge on the problem. In fact, real–world classification problems often are composed of datasets with labeled and unlabeled data [11, 12]: for this reason an interesting challenge is finding a way to exploit the unlabeled data for obtaining tighter bounds and, therefore, better error estimations.

In this paper, we present two methods for exploiting the unlabeled data in the Rademacher Complexity theory [2]. First, we show how the unlabeled data can have a role in reducing the confidence

term, by obtaining a new bound that takes into account both labeled and unlabeled data. Then, we propose a method, based on [7], which exploits the unlabeled data for selecting a better hypothesis space, which the classifier belongs to, resulting in a much sharper and accurate bound.

## 2 Theoretical framework and results

We consider the following prediction problem: based on a random observation of $X \in \mathcal{X} \subseteq \mathbb{R}^d$ one has to estimate $Y \in \mathcal{Y} \subseteq \{-1, 1\}$ by choosing a suitable prediction rule $f : X \to [-1, 1]$. The generalization error $L(f) = \mathbb{E}_{\{\mathcal{X}, \mathcal{Y}\}} \ell(f(X), Y)$ associated to the prediction rule is defined through a bounded loss function $\ell(f(X), Y) : [-1, 1] \times \mathcal{Y} \to [0, 1]$. We observe a set of labeled samples $\mathcal{D}_{n_l} : \{(X_1^l, Y_1^l), \cdots, (X_{n_l}^l, Y_{n_l}^l)\}$ and a set of unlabeled ones $\mathcal{D}_{n_u} : \{(X_1^u, \cdots, (X_{n_u}^u)\}$. The data consist of a sequence of independent, identically distributed (*i.i.d.*) samples with the same distribution $P(\mathcal{X}, \mathcal{Y})$ for $\mathcal{D}_{n_l}$ and $\mathcal{D}_{n_u}$. The goal is to obtain a bound on $L(f)$ that takes into account both the labeled and unlabeled data. As we do not know the distribution that have generated the data, we do not know $L(f)$ but only its empirical estimation $L_{n_l}(f) = 1/n_l \sum_{i=1}^{n_l} \ell(f(X_i^l), Y_i^l)$. In the typical context of *Structural Risk Minimization* (SRM) [13] we define an infinite sequence of hypothesis spaces of increasing complexity $\{\mathcal{F}_i, \ i = 1, 2, \cdots\}$, then we choose a suitable function space $\mathcal{F}_i$ and, consequently, a model $f^* \in \mathcal{F}_i$ that fits the data. As we do not know the true data distribution, we can only say that:

$$\{L(f) - L_{n_l}(f)\}_{f \in \mathcal{F}_i} \leq \sup_{f \in \mathcal{F}_i} \{L(f) - L_{n_l}(f)\} \tag{1}$$

or, equivalently:

$$L(f) \leq L_{n_l}(f) + \sup_{f \in \mathcal{F}_i} \{L(f) - L_{n_l}(f)\}, \quad \forall f \in \mathcal{F}_i \tag{2}$$

In this framework, the SRM procedure brings us to the following choice of the function space and the corresponding optimal classifier:

$$f^*, \mathcal{F}^* : \quad \arg \min_{\mathcal{F}_i \in \{\mathcal{F}_1, \mathcal{F}_2, \cdots\}} \left[ \min_{f \in \mathcal{F}_i} L_{n_l}(f)_{f \in \mathcal{F}_i} + \sup_{f \in \mathcal{F}_i} \{L(f) - L_{n_l}(f)\} \right] \tag{3}$$

Since the *generalization bias* ($\sup_{f \in \mathcal{F}_i} \{L(f) - L_{n_l}(f)\}$) is a random variable, it is possible to statistically analyze it and obtain a bound that holds with high probability [5].

From this point, we will consider two types of prediction rule with the associated loss function:

$$f_H(\boldsymbol{x}) = \text{sign}(\boldsymbol{w}^T \phi(\boldsymbol{x}) + b), \qquad \qquad \ell_H(f_H(\boldsymbol{x}), y) = \frac{1 - y f_H(\boldsymbol{x})}{2} \tag{4}$$

$$f_S(\boldsymbol{x}) = \begin{cases} \min(1, \boldsymbol{w}^T \phi(\boldsymbol{x}) + b) & \text{if } \boldsymbol{w}^T \phi(\boldsymbol{x}) + b > 0 \\ \max(-1, \boldsymbol{w}^T \phi(\boldsymbol{x}) + b) & \text{if } \boldsymbol{w}^T \phi(\boldsymbol{x}) + b \leq 0 \end{cases}, \quad \ell_S(f_S(\boldsymbol{x}), y) = \frac{1 - y f_S(\boldsymbol{x})}{2} \tag{5}$$

where $\phi(\cdot) : \mathbb{R}^d \to \mathbb{R}^D$ with $D >> d$, $\boldsymbol{w} \in \mathbb{R}^D$ and $b \in \mathbb{R}$. The function $\phi(\cdot)$ is introduced to allow for a later introduction of kernels, even though, for simplicity, we will focus only on the linear case. Note that both the *hard loss* $\ell_H(f_H(\boldsymbol{x}), y)$ and the *soft loss* (or *ramp loss*) [14] $\ell_S(f_S(\boldsymbol{x}), y)$ are bounded ($[0, 1]$) and symmetric ($\ell(f(\boldsymbol{x}), y) = 1 - \ell(f(\boldsymbol{x}), -y)$). Then, we recall the definition of *Rademacher Complexity* ($\mathcal{R}$) for a class of functions $\mathcal{F}$:

$$\hat{\mathcal{R}}_{n_l}(\mathcal{F}) = \mathbb{E}_\sigma \sup_{f \in \mathcal{F}} \frac{2}{n_l} \sum_{i=1}^{n_l} \sigma_i \ell(f(\boldsymbol{x}_i), y_i) = \mathbb{E}_\sigma \sup_{f \in \mathcal{F}} \frac{1}{n_l} \sum_{i=1}^{n_l} \sigma_i f(\boldsymbol{x}_i) \tag{6}$$

where $\sigma_1, \ldots, \sigma_{n_l}$ are $n_l$ independent Rademacher random variables, i.e. independent random variables for which $\mathbb{P}(\sigma_i = +1) = \mathbb{P}(\sigma_i = -1) = 1/2$, and the last equality holds if we use one of the losses defined before. Note that $\hat{\mathcal{R}}$ is a computable realization of the expected Rademacher Complexity $\mathcal{R}(\mathcal{F}) = \mathbb{E}_{(\mathcal{X}, \mathcal{Y})} \hat{\mathcal{R}}(\mathcal{F})$. The most renowed result in Rademacher Complexity theory states that [2]:

$$L(f)_{f \in \mathcal{F}} \leq L_{n_l}(f)_{f \in \mathcal{F}} + \hat{\mathcal{R}}_{n_l}(\mathcal{F}) + 3\sqrt{\frac{\log\left(\frac{2}{\delta}\right)}{2n_l}} \tag{7}$$

which holds with probability $(1 - \delta)$ and allows to solve the problem of Eq. (3).

## 2.1 Exploiting unlabeled samples for reducing the confidence term

Assuming that the amount of unlabeled data is larger than the number of labeled samples, we split them in blocks of similar size by defining the quantity $m = \lfloor n_u/n_l \rfloor + 1$, so that we can consider a ghost sample $D'_{mn_l}$ composed of $mn_l$ pattern. Then, we can upper bound the expected generalization bias in the following way [2]:

$$
\mathbb{E}_{\{\mathcal{X},\mathcal{Y}\}} \sup_{f \in \mathcal{F}} \{L(f) - L_{n_l}(f)\} = \mathbb{E}_{\{\mathcal{X},\mathcal{Y}\}} \sup_{f \in \mathcal{F}} \left[ \mathbb{E}_{\{\mathcal{X}',\mathcal{Y}'\}} \left[ \frac{1}{m} \sum_{i=1}^{m} \frac{1}{n_l} \sum_{k=(i-1)\cdot n_l+1}^{i\cdot n_l} \ell'_k \right] - \frac{1}{n_l} \sum_{i=1}^{n_l} \ell_i \right]
$$

$$
\leq \mathbb{E}_{\{\mathcal{X},\mathcal{Y}\}} \mathbb{E}_{\{\mathcal{X}',\mathcal{Y}'\}} \frac{1}{m} \sum_{i=1}^{m} \sup_{f \in \mathcal{F}} \left[ \frac{1}{n_l} \sum_{k=(i-1)\cdot n_l+1}^{i\cdot n_l} \left( \ell'_k - \ell_{|k|_{n_l}} \right) \right]
$$

$$
= \mathbb{E}_{\{\mathcal{X},\mathcal{Y}\}} \mathbb{E}_{\{\mathcal{X}',\mathcal{Y}'\}} \mathbb{E}_{\sigma} \frac{1}{m} \sum_{i=1}^{m} \sup_{f \in \mathcal{F}} \left[ \frac{1}{n_l} \sum_{k=(i-1)\cdot n_l+1}^{i\cdot n_l} \sigma_{|k|_{n_l}} \left[ \ell'_k - \ell_{|k|_{n_l}} \right] \right]
$$

$$
\leq \mathbb{E}_{\{\mathcal{X},\mathcal{Y}\}} \mathbb{E}_{\sigma} \frac{1}{m} \sum_{i=1}^{m} \sup_{f \in \mathcal{F}} \left[ \frac{2}{n_l} \sum_{k=(i-1)\cdot n_l+1}^{i\cdot n_l} \sigma_{|k|_{n_l}} \ell_k \right] = \mathbb{E}_{\{\mathcal{X},\mathcal{Y}\}} \frac{1}{m} \sum_{i=1}^{m} \hat{\mathcal{R}}^i_{n_l}(\mathcal{F})
$$

where $|k|_{n_l} = (k-1) \bmod (n_l) + 1$. The last quantity (that we call *Expected Extended Rademacher Complexity* $\mathbb{E}_{\{\mathcal{X},\mathcal{Y}\}} \hat{\mathcal{R}}_{n_u}(\mathcal{F})$) and the expected generalization bias are both deterministic quantities and we know only one realization of them, dependent on the sample. Then, we can use the McDiarmid's inequality [15] to obtain:

$$
\mathbb{P}\left[ \sup_{f \in \mathcal{F}} \{L(f) - L_{n_l}(f)\} \geq \hat{\mathcal{R}}_{n_u}(\mathcal{F}) + \epsilon \right] \leq \tag{8}
$$

$$
\mathbb{P}\left[ \sup_{f \in \mathcal{F}} \{L(f) - L_{n_l}(f)\} \geq \mathbb{E}_{\{\mathcal{X},\mathcal{Y}\}} \sup_{f \in \mathcal{F}} \{L(f) - L_{n_l}(f)\} + a\epsilon \right] + \tag{9}
$$

$$
\mathbb{P}\left[ \mathbb{E}_{\{\mathcal{X},\mathcal{Y}\}} \hat{\mathcal{R}}_{n_u}(\mathcal{F}) \geq \hat{\mathcal{R}}_{n_u}(\mathcal{F}) + (1-a)\epsilon \right] \leq \tag{10}
$$

$$
e^{-2n_l a^2 \epsilon^2} + e^{-\frac{(mn_l)}{2}(1-a)^2 \epsilon^2} \tag{11}
$$

with $a \in [0,1]$. By choosing $a = \frac{\sqrt{m}}{2+\sqrt{m}}$, we can write:

$$
\mathbb{P}\left[ \sup_{f \in \mathcal{F}} \{L(f) - L_{n_l}(f)\} \geq \frac{1}{m} \sum_{i=1}^{m} \hat{\mathcal{R}}^i_{n_l}(\mathcal{F}) + \epsilon \right] \leq 2e^{-\frac{2mn_l \epsilon^2}{(2+\sqrt{m})^2}} \tag{12}
$$

and obtain an explicit bound which holds with probability $(1-\delta)$:

$$
L(f)_{f \in \mathcal{F}} \leq L_{n_l}(f)_{f \in \mathcal{F}} + \frac{1}{m} \sum_{i=1}^{m} \hat{\mathcal{R}}^i_{n_l}(\mathcal{F}) + \frac{2+\sqrt{m}}{\sqrt{m}} \sqrt{\frac{\log\left(\frac{2}{\delta}\right)}{2n_l}} \tag{13}
$$

where $\hat{\mathcal{R}}^i_{n_l}(\mathcal{F})$ is the Rademacher Complexity of the class $\mathcal{F}$ computed on the $i$-th block of unlabeled data. Note that for $m = 1$ the training set does not contain any unlabeled data and the bound given by Eq. (3) is recovered, while for large $m$ the confidence term is reduced by a factor of 3. At a first sight, it would seem impossible to compute the term $\hat{\mathcal{R}}^i_{n_l}$ without knowing the labels of the data, but it is easy to show that this is not the case. In fact, let us define $\mathcal{K}^+_i = \left\{ k \in \{k = (i-1)\cdot n_l+1, \ldots, i\cdot n_l\} : \sigma_{|k|_{n_l}} = +1 \right\}$ and $\mathcal{K}^-_i =$

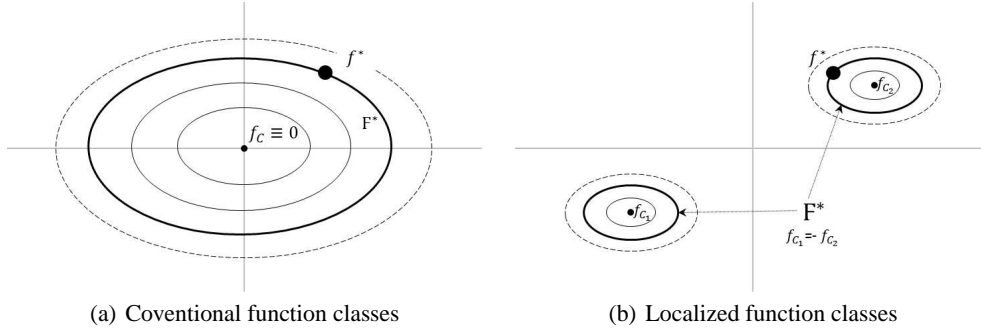

(a) Coventional function classes          (b) Localized function classes

Figure 1: The effect of selecting a better center for the hypothesis classes.

$\left\{ k \in \{k = (i-1) \cdot n_l + 1, \ldots, i \cdot n_l\} : \sigma_{|k|_{n_l}} = -1 \right\}$, then we have:

$$\hat{\mathcal{R}}_{n_u}(\mathcal{F}) = 1 + \frac{1}{m} \sum_{i=1}^{m} \mathbb{E}_\sigma \sup_{f \in \mathcal{F}} \frac{2}{n_l} \left[ \sum_{k \in \mathcal{K}_i^+} \ell(f_k, y_k) - \sum_{k \in \mathcal{K}_i^-} \ell(f_k, y_k) - \sum_{k \in \mathcal{K}_i^+} 1 \right]$$

$$= 1 + \frac{1}{m} \sum_{i=1}^{m} \mathbb{E}_\sigma \sup_{f \in \mathcal{F}} \left[ -\frac{2}{n_l} \sum_{k \in \mathcal{K}_i^+} \ell(f_k, -y_k) - \frac{2}{n_l} \sum_{k \in \mathcal{K}_i^-} \ell(f_k, y_k) \right]$$

$$= 1 + \frac{1}{m} \sum_{i=1}^{m} \mathbb{E}_\sigma \sup_{f \in \mathcal{F}} \left[ -\frac{2}{n_l} \sum_{k=(i-1)\cdot n_l + 1}^{i \cdot n_l} \ell(f_k, -\sigma_{|k|_{n_l}} y_k) \right]$$

$$= 1 - \frac{1}{m} \sum_{i=1}^{m} \mathbb{E}_\sigma \inf_{f \in \mathcal{F}} \left[ \frac{2}{n_l} \sum_{k=(i-1)\cdot n_l + 1}^{i \cdot n_l} \ell(f_k, \sigma_{|k|_{n_l}}) \right]$$

which corresponds to solving a classification problem using all the available data with random labels. The expectation can be easily computed with some Monte Carlo trials.

## 2.2    Exploiting the unlabeled data for tightening the bound

Another way of exploiting the unlabeled data is to use them for selecting a more suitable sequence of hypothesis spaces. For this purpose we could use some of the unlabeled samples or, even better, the $n_c = n_u - \lfloor n_u/n_l \rfloor n_l$ samples left from the procedure of the previous section. The idea is inspired by the work of [3] and [7], which propose to inflate the hypothesis classes by centering them around a 'good' classifier. Usually, in fact, we have no a-priori information on what can be considered a good choice of the class center, so a natural choice is the origin [13], as in Figure 1(a). However, if it happens that the center is 'close' to the optimal classifier, the search for a suitable class will stop very soon and the resulting Rademacher Complexity will be consequently reduced (see Figure 1(b)). We propose here a method for finding two possible 'good' centers for the hypothesis classes. Let us consider $n_c$ unlabeled samples and run a clustering algorithm on them, by setting the number of clusters to 2, and obtaining two clusters $C_1$ and $C_2$. We build two distinct labeled datasets by assigning the labels $+1$ and $-1$ to $C_1$ and $C_2$, respectively, and then vice-versa. Finally, we build two classifiers $f_{C_1}(\boldsymbol{x})$ and $f_{C_2}(\boldsymbol{x}) = -f_{C_1}(\boldsymbol{x})$ by learning the two datasets[3]. The two classifiers, which have been found using only unlabeled samples, can then be used as centers for searching a better hypothesis class. It is worthwhile noting that any supervised learning algorithm can be used [16], because the centers are only a hint for a better centered hypothesis space: their actual classification performance is not of paramount importance. The underlying principle that inspired

this procedure relies on the reasonable hypothesis that $\mathbb{P}(\mathcal{X})$ is correlated with $\mathbb{P}(\mathcal{X}, \mathcal{Y})$: in fact, in an unlucky scenario, where the two classes are heavily overlapped, the method would obviously fail.

Choosing a good center for the SRM procedure can greatly reduce the second term of the bound given by Eq. (13) [7] (the bias or complexity term). Note, however, that the confidence term is not affected, so we propose here an improved bound, which makes this term depending on $\hat{\mathcal{R}}_{n_l}^i(\mathcal{F})$ as well. We use a recent concentration result for *Self Bounding Functions* [17], instead of the looser McDiarmid's inequality. The detailed proof is omitted due to space constraints and we give here only the sketch (it is a more general version of the proof in [18] for Rademacher Complexities):

$$\mathbb{P}\left[\sup_{f \in \mathcal{F}}\{L(f) - L_{n_l}(f)\} \geq \hat{\mathcal{R}}_{n_u}(\mathcal{F}) + \epsilon\right] \leq e^{-2n_l a^2 \epsilon^2} + e^{-\frac{(mn_l)(1-a)^2\epsilon^2}{2\mathbb{E}_{\{\mathcal{X},\mathcal{Y}\}}\hat{\mathcal{R}}_{n_u}(\mathcal{F})}} \qquad (14)$$

with $a \in [0,1]$. Choosing $a = \frac{\sqrt{m}}{\sqrt{m}+2\sqrt{\mathbb{E}_{\{\mathcal{X},\mathcal{Y}\}}\frac{1}{m}\sum_{i=1}^m \hat{\mathcal{R}}_{n_l}^i(\mathcal{F})}}$, we obtain:

$$\mathbb{P}\left[\sup_{f \in \mathcal{F}}\{L(f) - L_{n_l}(f)\} \geq \hat{\mathcal{R}}_{n_u}(\mathcal{F}) + \epsilon\right] \leq 2e^{-\frac{2mn_l\epsilon^2}{\left(\sqrt{m}+2\sqrt{\mathbb{E}_{\{\mathcal{X},\mathcal{Y}\}}\hat{\mathcal{R}}_{n_u}(\mathcal{F})}\right)^2}} \qquad (15)$$

so that the following explicit bound holds with probability $(1-\delta)$:

$$L(f)_{f \in \mathcal{F}} \leq L_{n_l}(f)_{f \in \mathcal{F}} + \hat{\mathcal{R}}_{n_u}(\mathcal{F}) + \frac{2\sqrt{\mathbb{E}_{\{\mathcal{X},\mathcal{Y}\}}\hat{\mathcal{R}}_{n_u}(\mathcal{F})} + \sqrt{m}}{\sqrt{m}}\sqrt{\frac{\log\left(\frac{2}{\delta}\right)}{2n_l}} \qquad (16)$$

Note that, in the worst case, $\mathbb{E}_{\{\mathcal{X},\mathcal{Y}\}}\hat{\mathcal{R}}_{n_u}(\mathcal{F}) = 1$ and we obtain again Eq. (13). Unfortunately, the Expected Extended Rademacher Complexity cannot be computed, but we can upper bound it with its empirical version (see, for example, [19], pages 420–422, for a justificaton of this step) as in Eq.(10) to obtain:

$$\mathbb{P}\left[\sup_{f \in \mathcal{F}}\{L(f) - L_{n_l}(f)\} \geq \hat{\mathcal{R}}_{n_u}(\mathcal{F}) + \epsilon\right] \leq e^{-2n_l a^2 \epsilon^2} + e^{-\frac{(mn_l)(1-a)^2\epsilon^2}{2(\hat{\mathcal{R}}_{n_u}(\mathcal{F})+(1-a)\epsilon)}} \qquad (17)$$

with $a \in [0,1]$. Differently from Eq. (15) the previous expression cannot be put in explicit form, but it can be simply computed numerically by writing it as:

$$L(f)_{f \in \mathcal{F}} \leq L_{n_l}(f)_{f \in \mathcal{F}} + \frac{1}{m}\sum_{i=1}^m \hat{\mathcal{R}}_{n_l}^i(\mathcal{F}) + \epsilon_u^b \qquad (18)$$

The value $\epsilon_u^b$ can be obtained by upper bounding with $\delta$ the last term of Eq. (17) and solving the inequality respect to $a$ and $\epsilon$, so that the bound holds with probability $(1-\delta)$.

We can show the improvements obtained through these new results, by plotting the values of the confidence terms and comparing them with the conventional one [2]. Figure 2 shows the value of $\epsilon_l$ in Eq. (7) against $\epsilon_u$, the corresponding term in Eq. (13), and $\epsilon_u^b$, as a function of the number of samples.

## 3  Performing the Structural Risk Minimization procedure

Computing the values of the bounds described in the previous sections is a straightforward process, at least in theory. The empirical error $L_{n_l}(f)$ is found by learning a classifier with the original labeled dataset, while the (Extended) Rademacher Complexity $\hat{\mathcal{R}}_{n_l}^i(\mathcal{F})$ is computed by learning the dataset composed of both labeled and unlabeled samples with random labels.

In order apply in practice the results of the previous section and to better control the hypothesis space, we formulate the learning phase of the classifier as the following optimization problem, based

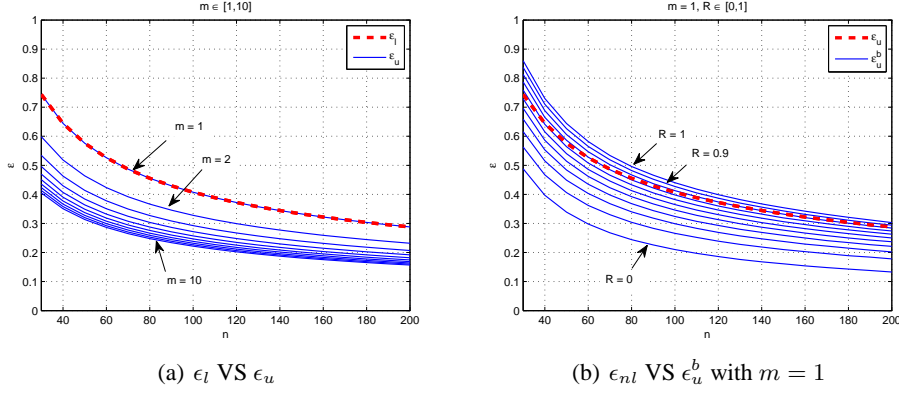

(a) $\epsilon_l$ VS $\epsilon_u$        (b) $\epsilon_{nl}$ VS $\epsilon_u^b$ with $m = 1$

Figure 2: Comparison of the new confidence terms with the conventional one.

on the Ivanov version of the Support Vector Machine (I-SVM) [13]:

$$\min_{\boldsymbol{w},b,\boldsymbol{\xi}} \quad \sum_{i=1}^{n} \eta_i \tag{19}$$
$$\|\boldsymbol{w} - \hat{\boldsymbol{w}}\|^2 \leq \rho^2$$
$$y_i\left(\boldsymbol{w}^T\phi(\boldsymbol{x}_i) + b\right) \geq 1 - \xi_i$$
$$\xi_i \geq 0, \quad \eta_i = \min\left(2, \xi_i\right)$$

where the size of the hypothesis space, centered in $\hat{\boldsymbol{w}}$, is controlled by the hyperparameter $\rho$ and the last constraint is introduced for bounding the SVM loss function, which would be otherwise unbounded and would prevent the application of the theory developed so far. Note that, in practice, two sub-problems must be solved: the first one with $\hat{\boldsymbol{w}} = +\hat{\boldsymbol{w}}_{C_1}$ and the second one with $\hat{\boldsymbol{w}} = -\hat{\boldsymbol{w}}_{C_1}$, then the solution corresponding to the smaller value of the objective function is selected.

Unfortunately, solving a classification problem with a bounded loss function is computationally intractable, because the problem is no longer convex and even state-of-the-art solvers like, for example, CPLEX [20] fail to found an exact solution, when the training set size exceeds few tens of samples. Therefore, we propose here to find an approximate solution through well–known algorithms like, for example, the Peeling [6] or the Convex–Concave Constrained Programming (CCCP) technique [14, 21, 22]. Furthermore, we derive a dual formulation of problem (19) that allows us exploiting the well known Sequential Minimal Optimization (SMO) algorithm for SVM learning [23].

Problem (19) can be rewritten in the equivalent Tikhonov formulation:

$$\min_{\boldsymbol{w},b,\boldsymbol{\xi}} \quad \frac{1}{2}\|\boldsymbol{w} - \hat{\boldsymbol{w}}\|^2 + C\sum_{i=1}^{n} \eta_i \tag{20}$$
$$y_i\left(\boldsymbol{w}^T\phi(\boldsymbol{x}_i) + b\right) \geq 1 - \xi_i$$
$$\xi_i \geq 0, \quad \eta_i = \min\left(2, \xi_i\right)$$

which gives the same solution of the Ivanov formulation for some value of $C$ [13]. The method for finding the value of $C$, corresponding to a given value of $\rho$, is reported in [10], where it is also shown that $C$ cannot be used directly to control the hypothesis space. Then, it is possible to apply the CCCP technique, which is synthesized in Algorithm 1, by splitting the objective function in its convex and concave parts:

$$\min_{\boldsymbol{w},b,\boldsymbol{\xi}} \overbrace{\frac{1}{2}\|\boldsymbol{w} - \hat{\boldsymbol{w}}\|^2 + C\sum_{i=1}^{n}\xi_i}^{\mathcal{J}_{\text{convex}}(\boldsymbol{\theta})} \overbrace{-C\sum_{i=1}^{n}\varsigma_i}^{\mathcal{J}_{\text{concave}}(\boldsymbol{\theta})} \tag{21}$$
$$y_i\left(\boldsymbol{w}^T\phi(\boldsymbol{x}_i) + b\right) \geq 1 - \xi_i$$
$$\xi_i \geq 0, \quad \varsigma_i = \max(0, \xi_i - 2)$$

where $\boldsymbol{\theta} = [\boldsymbol{w}|b]$ is introduced to simplify the notation. Obviously, the algorithm does not guarantee to find the optimal solution, but it converges to a (usually good) solution in a finite number of steps [14]. To apply the algorithm we must compute the derivative of the concave part of the objective function:

$$\left( \frac{d\mathcal{J}_{\text{concave}}(\boldsymbol{\theta})}{d\boldsymbol{\theta}} \bigg|_{\boldsymbol{\theta}^t} \right) \boldsymbol{\theta} = \left( \sum_{i=1}^{n} \frac{d\left(-C\varsigma_i\right)}{d\boldsymbol{\theta}} \bigg|_{\boldsymbol{\theta}^t} \right) \boldsymbol{\theta} = \sum_{i=1}^{n} \beta_i y_i \left( \boldsymbol{w}^T \phi(\boldsymbol{x}_i) + b \right) \tag{22}$$

Then, the learning problem becomes:

$$\min_{\boldsymbol{w},b,\boldsymbol{\xi}} \frac{1}{2} \|\boldsymbol{w} - \hat{\boldsymbol{w}}\|^2 + C \sum_{i=1}^{n} \xi_i + \sum_{i=1}^{n} \Delta_i y_i \left( \boldsymbol{w}^T \phi(\boldsymbol{x}_i) + b \right) \tag{23}$$

$$y_i \left( \boldsymbol{w}^T \phi(\boldsymbol{x}_i) + b \right) \geq 1 - \xi_i, \quad \xi_i \geq 0$$

where

$$\Delta_i = \begin{cases} C & \text{if } y_i f^t(x_t) < -1 \\ 0 & \text{otherwise} \end{cases} \tag{24}$$

Finally, it is possible to obtain the dual formulation (derivation is omitted due to lack of space):

$$\min_{\boldsymbol{\beta}} \frac{1}{2} \sum_{i=1}^{n} \sum_{j=1}^{n} \beta_i \beta_j y_i y_j K(\boldsymbol{x}_i, \boldsymbol{x}_j) + \sum_{i=1}^{n} \left[ \sum_{j=1}^{n_{C_1}} \hat{\alpha}_j y_i \hat{y}_j K(\hat{\boldsymbol{x}}_j, \boldsymbol{x}_i) - 1 \right] \beta_i \tag{25}$$

$$-\Delta_i \leq \beta_i \leq C - \Delta_i, \quad \sum_{i=1}^{n} \beta_i y_i = 0$$

where we have used the kernel trick [24] $K(\cdot, \cdot) = \phi(\cdot)^T \phi(\cdot)$.

## 4   A case study

We consider the MNIST dataset [25], which consists of 62000 images, representing the numbers from 0 to 9: in particular, we consider the 13074 patterns containing 0's and 1's, allowing us to deal with a binary classification problem. We simulate the small–sample regime by randomly sampling a training set with low cardinality ($n_l < 500$), while the remaining $13074 - n_l$ images are used as a test set or as an unlabeled dataset, by simply discarding the labels. In order to build statistically relevant results, this procedure is repeated 30 times.

In Table 1 we compare the conventional bound with our proposal. In the first column the number of labeled patterns ($n_l$) is reported, while the second column shows the number of unlabeled ones ($n_u$). The optimal classifier $f^*$ is selected by varying $\rho$ in the range $[10^{-6}, 1]$, and selecting the function corresponding to the minimum of the generalization error estimate provided by each bound. Then, for each case, the selected $f^*$ is tested on the remaining $13074 - (n_l + n_u)$ samples and the classification results are reported in column three and four, respectively. The results show that the $f^*$ selected by exploiting the unlabeled patterns behaves better than the other and, furthermore, the estimated $L(f)$, reported in column five and six, shows that the bound is tighter, as expected by theory.

The most interesting result, however, derives from the use of the new bound of Eq. (18), as reported in Table 2, where the unlabeled data is exploited for selecting a more suitable center of the hypothesis space. The results are reported analogously to Table 1. Note that, for each experiment, 30%

---

**Algorithm 1** CCCP procedure

    Initialize $\boldsymbol{\theta}^0$
   **repeat**
      $\boldsymbol{\theta}^{t+1} = \arg\min_{\boldsymbol{\theta}} \mathcal{J}_{\text{convex}}(\boldsymbol{\theta}) + \left( \frac{d\mathcal{J}_{\text{concave}}(\boldsymbol{\theta})}{d\boldsymbol{\theta}} \big|_{\boldsymbol{\theta}^t} \right) \boldsymbol{\theta}$
   **until** $\boldsymbol{\theta}^{t+1} = \boldsymbol{\theta}^t$

---

Table 1: Model selection and error estimation, exploiting unlabeled data for tightening the bound.

| $n_l$ | $n_u$ | Test error of $f^*$ Eq. (7) | Eq. (13) | Estimated $L(f)$ Eq. (7) | Eq. (13) |
|---|---|---|---|---|---|
| 10 | 20 | $13.20 \pm 0.86$ | $\mathbf{12.40 \pm 0.82}$ | $194.00 \pm 0.97$ | $\mathbf{157.70 \pm 0.97}$ |
| 20 | 40 | $\mathbf{8.93 \pm 1.20}$ | $\mathbf{8.93 \pm 1.29}$ | $142.00 \pm 1.06$ | $\mathbf{116.33 \pm 1.06}$ |
| 40 | 80 | $6.26 \pm 0.16$ | $\mathbf{6.02 \pm 0.17}$ | $103.00 \pm 0.59$ | $\mathbf{84.85 \pm 0.59}$ |
| 60 | 120 | $5.95 \pm 0.12$ | $\mathbf{5.88 \pm 0.13}$ | $85.50 \pm 0.48$ | $\mathbf{70.68 \pm 0.48}$ |
| 80 | 160 | $5.61 \pm 0.07$ | $\mathbf{5.30 \pm 0.07}$ | $73.70 \pm 0.40$ | $\mathbf{60.86 \pm 0.40}$ |
| 100 | 200 | $\mathbf{5.36 \pm 0.21}$ | $5.51 \pm 0.22$ | $66.10 \pm 0.37$ | $\mathbf{54.62 \pm 0.37}$ |
| 120 | 240 | $\mathbf{4.98 \pm 0.40}$ | $5.36 \pm 0.40$ | $61.30 \pm 0.33$ | $\mathbf{50.82 \pm 0.33}$ |
| 150 | 300 | $4.41 \pm 0.53$ | $\mathbf{4.08 \pm 0.51}$ | $55.10 \pm 0.28$ | $\mathbf{45.73 \pm 0.28}$ |
| 170 | 340 | $3.59 \pm 0.57$ | $\mathbf{3.40 \pm 0.64}$ | $52.40 \pm 0.26$ | $\mathbf{43.60 \pm 0.26}$ |
| 200 | 400 | $2.75 \pm 0.47$ | $\mathbf{2.67 \pm 0.48}$ | $48.10 \pm 0.19$ | $\mathbf{39.98 \pm 0.19}$ |
| 250 | 500 | $2.07 \pm 0.03$ | $\mathbf{2.05 \pm 0.03}$ | $42.70 \pm 0.22$ | $\mathbf{35.44 \pm 0.22}$ |
| 300 | 600 | $2.02 \pm 0.04$ | $\mathbf{1.94 \pm 0.04}$ | $39.20 \pm 0.17$ | $\mathbf{32.57 \pm 0.17}$ |
| 400 | 800 | $1.93 \pm 0.02$ | $\mathbf{1.79 \pm 0.02}$ | $34.90 \pm 0.19$ | $\mathbf{29.16 \pm 0.19}$ |

Table 2: Model selection and error estimation, exploiting unlabeled data for selecting a more suitable hypothesis center.

| $n_l$ | $n_u$ | Test error of $f^*$ Eq. (7) | Eq. (18) | Estimated $L(f)$ Eq. (7) | Eq. (18) |
|---|---|---|---|---|---|
| 7 | 3 | $13.20 \pm 0.86$ | $\mathbf{8.98 \pm 1.12}$ | $219.15 \pm 0.97$ | $\mathbf{104.01 \pm 1.62}$ |
| 14 | 6 | $8.93 \pm 1.20$ | $\mathbf{5.10 \pm 0.67}$ | $159.79 \pm 1.06$ | $\mathbf{86.70 \pm 0.01}$ |
| 28 | 12 | $6.26 \pm 0.16$ | $\mathbf{3.05 \pm 0.23}$ | $115.58 \pm 0.59$ | $\mathbf{51.35 \pm 0.00}$ |
| 42 | 18 | $5.95 \pm 0.12$ | $\mathbf{2.36 \pm 0.23}$ | $95.77 \pm 0.48$ | $\mathbf{38.37 \pm 0.00}$ |
| 56 | 24 | $5.61 \pm 0.07$ | $\mathbf{1.96 \pm 0.14}$ | $82.59 \pm 0.40$ | $\mathbf{31.39 \pm 0.00}$ |
| 70 | 30 | $5.36 \pm 0.21$ | $\mathbf{1.63 \pm 0.11}$ | $74.05 \pm 0.37$ | $\mathbf{26.83 \pm 0.00}$ |
| 84 | 36 | $4.98 \pm 0.40$ | $\mathbf{1.44 \pm 0.11}$ | $68.56 \pm 0.33$ | $\mathbf{23.77 \pm 0.00}$ |
| 105 | 45 | $4.41 \pm 0.53$ | $\mathbf{1.27 \pm 0.09}$ | $61.59 \pm 0.28$ | $\mathbf{20.36 \pm 0.00}$ |
| 119 | 51 | $3.59 \pm 0.57$ | $\mathbf{1.20 \pm 0.08}$ | $58.50 \pm 0.26$ | $\mathbf{18.77 \pm 0.00}$ |
| 140 | 60 | $2.75 \pm 0.47$ | $\mathbf{1.08 \pm 0.09}$ | $53.72 \pm 0.19$ | $\mathbf{16.82 \pm 0.00}$ |
| 175 | 75 | $2.07 \pm 0.03$ | $\mathbf{0.92 \pm 0.05}$ | $47.73 \pm 0.22$ | $\mathbf{14.52 \pm 0.00}$ |
| 210 | 90 | $2.02 \pm 0.04$ | $\mathbf{0.81 \pm 0.07}$ | $43.79 \pm 0.17$ | $\mathbf{12.91 \pm 0.00}$ |
| 280 | 120 | $1.93 \pm 0.02$ | $\mathbf{0.70 \pm 0.06}$ | $38.88 \pm 0.19$ | $\mathbf{10.86 \pm 0.00}$ |

of the data ($n_u$) are used for selecting the hypothesis center and the remaining ones ($n_l$) are used for training the classifier. The proposed method consistently selects a better classifier, which registers a threefold classification error reduction on the test set, especially for training sets of smaller cardinality. The estimation of $L(f)$ is largely reduced as well.

We have to consider that this very clear performance increase is also favoured by the characteristics of the MNIST dataset, which consists of well–separated classes: this particular data distribution implies that only few samples suffice for identifying a good hypothesis center. Many more experiments with different datasets and varying the ratio between labeled and unlabeled samples are needed, and are currently underway, for establishing the general validity of our proposal but, in any case, these results appear to be very promising.

## 5  Conclusion

In this paper we have studied two methods which exploit unlabeled samples to tighten the Rademacher Complexity bounds on the generalization error of linear (kernel) classifiers. The first method improves a very well–known result, while the second one aims at changing the entire approach by selecting more suitable hypothesis spaces, not only acting on the bound itself. The recent literature on the theory of bounds attempts to obtain tighter bounds through more refined concentration inequalities (e.g. improving Mc Diarmid's inequality), but we believe that the idea of reducing the size of the hypothesis space is a more appealing field of research because it opens the road to possible significant improvements.

## Footnotes

[1]See, for example, the NIPS 2004 Workshop *(Ab)Use of Bounds* or the 2002 Neurocolt Workshop on *Bounds less than 0.5*

[2] we define $\ell(f(\boldsymbol{x}_i), y_i) \equiv \ell_i$ to simplify the notation

[3]Note that we could build only one classifier by assigning the most probable labels to the $n_c$ samples, according to the $n_l$ labeled ones but, rigorously speaking, this is not allowed by the SRM principle, because it would lead to use the same data for both choosing the space of functions and computing the Rademacher Complexity.

## References

[1] V.N. Vapnik and A.Y. Chervonenkis. On the uniform convergence of relative frequencies of events to their probabilities. *Theory of Probability and its Applications*, 16:264, 1971.

[2] P.L. Bartlett and S. Mendelson. Rademacher and Gaussian complexities: Risk bounds and structural results. *The Journal of Machine Learning Research*, 3:463–482, 2003.

[3] P.L. Bartlett, O. Bousquet, and S. Mendelson. Local rademacher complexities. *The Annals of Statistics*, 33(4):1497–1537, 2005.

[4] O. Bousquet and A. Elisseeff. Stability and generalization. *The Journal of Machine Learning Research*, 2:499–526, 2002.

[5] P.L. Bartlett, S. Boucheron, and G. Lugosi. Model selection and error estimation. *Machine Learning*, 48(1):85–113, 2002.

[6] D. Anguita, A. Ghio, and S. Ridella. Maximal discrepancy for support vector machines. *Neurocomputing*, 74(9):1436–1443, 2011.

[7] D. Anguita, A. Ghio, L. Oneto, and S. Ridella. Selecting the Hypothesis Space for Improving the Generalization Ability of Support Vector Machines. In *The 2011 International Joint Conference on Neural Networks (IJCNN), San Jose, California*. IEEE, 2011.

[8] S. Arlot and A. Celisse. A survey of cross-validation procedures for model selection. *Statistics Surveys*, 4:40–79, 2010.

[9] B. Efron and R. Tibshirani. *An introduction to the bootstrap*. Chapman & Hall/CRC, 1993.

[10] D. Anguita, A. Ghio, L. Oneto, and S. Ridella. In-sample Model Selection for Support Vector Machines. In *The 2011 International Joint Conference on Neural Networks (IJCNN), San Jose, California*. IEEE, 2011.

[11] K.P. Bennett and A. Demiriz. Semi-supervised support vector machines. In *Advances in neural information processing systems 11: proceedings of the 1998 conference*, page 368. The MIT Press, 1999.

[12] O. Chapelle, B. Scholkopf, and A. Zien. Semi-supervised learning. *The MIT Press*, page 528, 2010.

[13] V.N. Vapnik. *The nature of statistical learning theory*. Springer Verlag, 2000.

[14] R. Collobert, F. Sinz, J. Weston, and L. Bottou. Trading convexity for scalability. In *Proceedings of the 23rd international conference on Machine learning*, pages 201–208. ACM, 2006.

[15] C. McDiarmid. On the method of bounded differences. *Surveys in combinatorics*, 141(1):148–188, 1989.

[16] S. Haykin. *Neural networks: a comprehensive foundation*. Prentice Hall PTR Upper Saddle River, NJ, USA, 1994.

[17] S. Boucheron, G. Lugosi, and P. Massart. On concentration of self-bounding functions. *Electronic Journal of Probability*, 14:1884–1899, 2009.

[18] S. Boucheron, G. Lugosi, and P. Massart. Concentration inequalities using the entropy method. *The Annals of Probability*, 31(3):1583–1614, 2003.

[19] G. Casella and R.L. Berger. Statistical inference. 2001.

[20] I. CPLEX. 11.0 users manual. *ILOG SA*, 2008.

[21] J. Wang, X. Shen, and W. Pan. On efficient large margin semisupervised learning: Method and theory. *Journal of Machine Learning Research*, 10:719–742, 2009.

[22] J. Wang and X. Shen. Large margin semi–supervised learning. *Journal of Machine Learning Research*, 8:1867–1891, 2007.

[23] J. Platt. Sequential minimal optimization: A fast algorithm for training support vector machines. *Advances in Kernel MethodsSupport Vector Learning*, 208:1–21, 1998.

[24] J. Shawe-Taylor and N. Cristianini. Margin distribution and soft margin. *Advances in Large Margin Classifiers*, pages 349–358, 2000.

[25] H. Larochelle, D. Erhan, A. Courville, J. Bergstra, and Y. Bengio. An empirical evaluation of deep architectures on problems with many factors of variation. In *24th ICML*, pages 473–480, 2007.

